# Metric Learning with Multiple Kernels

**Jun Wang**      **Huyen Do**      **Adam Woznica**      **Alexandros Kalousis**

AI Lab, Department of Informatics
University of Geneva, Switzerland
{Jun.Wang, Huyen.Do, Adam.Woznica, Alexandros.Kalousis}@unige.ch

## Abstract

Metric learning has become a very active research field. The most popular representative–Mahalanobis metric learning–can be seen as learning a linear transformation and then computing the Euclidean metric in the transformed space. Since a linear transformation might not always be appropriate for a given learning problem, kernelized versions of various metric learning algorithms exist. However, the problem then becomes finding the appropriate kernel function. Multiple kernel learning addresses this limitation by learning a linear combination of a number of predefined kernels; this approach can be also readily used in the context of multiple-source learning to fuse different data sources. Surprisingly, and despite the extensive work on multiple kernel learning for SVMs, there has been no work in the area of metric learning with multiple kernel learning. In this paper we fill this gap and present a general approach for metric learning with multiple kernel learning. Our approach can be instantiated with different metric learning algorithms provided that they satisfy some constraints. Experimental evidence suggests that our approach outperforms metric learning with an unweighted kernel combination and metric learning with cross-validation based kernel selection.

## 1   Introduction

Metric learning (ML), which aims at learning dissimilarities by determining the importance of different input features and their correlations, has become a very active research field over the last years [23, 5, 3, 14, 22, 7, 12]. The most prominent form of ML is learning the Mahalanobis metric. Its computation can be seen as a two-step process; in the first step we perform a linear projection of the instances and in the second step we compute their Euclidean metric in the projected space.

Very often a linear projection cannot adequately represent the inherent complexities of a problem at hand. To address this limitation various works proposed kernelized versions of ML methods in order to implicitly compute a linear transformation and Euclidean metric in some non-linear feature space; this computation results in a non-linear projection and distance computation in the original input space [23, 5, 3, 14, 22]. However, we are now faced with a new problem, namely that of finding the appropriate kernel function and the associated feature space matching the requirements of the learning problem.

The simplest approach to address this problem is to select the best kernel from a predefined kernel set using internal cross-validation. The main drawback of this approach is that only one kernel is selected which limits the expressiveness of the resulting method. Additionally, this approach is limited to a small number of kernels–due to computational constraints–and requires the use of extra data. Multiple Kernel Learning (MKL) [10, 17] lifts the above limitations by learning a linear combination of a number of predefined kernels. The MKL approach can also naturally handle the multiple-source learning scenarios where instead of combining kernels defined on a single input data, which depending on the selected kernels could give rise to feature spaces with redundant

features, we combine different and complementary data sources. In [11, 13] the authors propose a method that learns a distance metric for multiple-source problems within a multiple-kernel scenario. The proposed method defines the distance of two instances as the sum of their distances in the feature spaces induced by the different kernels. During learning, a set of Mahalanobis metrics, one for each source, are learned together. However, this approach ignores the potential correlations between the different kernels. To the best of our knowledge most of the work on MKL has been confined in the framework of SVMs and despite the recent popularity of ML there exists so far no work that performs MKL in the ML framework by learning a distance metric in the weighted linear combination of feature spaces.

In this paper we show how to perform the Mahalanobis ML with MKL. We first propose a general framework of ML with MKL which can be instantiated with virtually any Mahalanobis ML algorithm $h$ provided that the latter satisfies some stated conditions. We examine two parametrizations of the learning problem that give rise to two alternative formulations, denoted by *ML$_h$-MKL$_\mu$* and *ML$_h$-MKL$_\mathbf{P}$*. Our approach can be seen as the counterpart of MKL with SVMs [10, 20, 17] for ML. Since the learned metric matrix has a regularized form (i.e. it has internal structure) we propose a straightforward non-regularized version of ML with MKL, denoted by *NR-ML$_h$-MKL*; however, due to the number of free parameters the non-regularized version can only scale with very small number of kernels and requires ML methods that are able to cope with large dimensionalities. We performed a number of experiments for ML with MKL in which, for the needs of this paper, we have chosen the well known Large Margin Nearest Neighbor [22] (*LMNN*) algorithm as the ML method $h$. The experimental results suggest that *LMNN-MKL$_\mathbf{P}$* outperforms *LMNN* with an unweighted kernel combination and the single best kernel selected by internal cross-validation.

## 2   Preliminaries

In the different flavors of metric learning we are given a matrix of learning instances $\mathbf{X} : n \times d$, the $i$-th row of which is the $\mathbf{x}_i^T \in \mathbb{R}^d$ instance, and a vector of class labels $\mathbf{y} = (y_1, \ldots, y_n)^T, y_i \in \{1, \ldots, c\}$. Consider a mapping $\mathbf{\Phi}_l(\mathbf{x})$ of instances $\mathbf{x}$ to some feature space $\mathcal{H}_l$, i.e. $\mathbf{x} \rightarrow \mathbf{\Phi}_l(\mathbf{x}) \in \mathcal{H}_l$. The corresponding kernel function $k_l(\mathbf{x}_i, \mathbf{x}_j)$ computes the inner product of two instances in the $\mathcal{H}_l$ feature space, i.e. $k_l(\mathbf{x}_i, \mathbf{x}_j) = \langle \mathbf{\Phi}_l(\mathbf{x}_i), \mathbf{\Phi}_l(\mathbf{x}_j) \rangle$. We denote dimensionality of $\mathcal{H}_l$ (possibly infinite) as $d_l$. The squared Mahalanobis distance of two instances in the $\mathcal{H}_l$ space is given by $d_{\mathbf{M}_l}^2(\mathbf{\Phi}_l(\mathbf{x}_i), \mathbf{\Phi}_l(\mathbf{x}_j)) = (\mathbf{\Phi}_l(\mathbf{x}_i) - \mathbf{\Phi}_l(\mathbf{x}_j))^T \mathbf{M}_l(\mathbf{\Phi}_l(\mathbf{x}_i) - \mathbf{\Phi}_l(\mathbf{x}_j))$, where $\mathbf{M}_l$ is a Positive Semi-Definite (PSD) metric matrix in the $\mathcal{H}_l$ space ($\mathbf{M}_l \succeq 0$). For some given ML method $h$ we optimize (most often minimize) some cost function $F_h$ with respect to the $\mathbf{M}_l$ metric matrix[1] under the PSD constraint for $\mathbf{M}_l$ and an additional set of pairwise distance constraints $\mathcal{C}_h(\{d_{\mathbf{M}_l}^2(\mathbf{\Phi}_l(\mathbf{x}_i), \mathbf{\Phi}_l(\mathbf{x}_j)) \mid i, j = 1, \ldots, n\})$ that depend on the choice of $h$, e.g. similarity and dissimilarity pairwise constraint [3] and relative comparison constraint [22]. In the reminder of this paper, for simplicity, we denote this set of constraints as $\mathcal{C}_h(d_{\mathbf{M}_l}^2(\mathbf{\Phi}_l(\mathbf{x}_i), \mathbf{\Phi}_l(\mathbf{x}_j)))$. The kernelized ML optimization problem can be now written as:

$$\min_{\mathbf{M}_l} F_h(\mathbf{M}_l) \quad s.t. \quad \mathcal{C}_h(d_{\mathbf{M}_l}^2(\mathbf{\Phi}_l(\mathbf{x}_i), \mathbf{\Phi}_l(\mathbf{x}_j))), \ \mathbf{M}_l \succeq 0 \tag{1}$$

Kernelized ML methods do not require to learn the explicit form of the Mahalanobis metric $\mathbf{M}_l$. It was shown in [9] that the optimal solution of the Mahalanobis metric $\mathbf{M}_l$ is in the form of $\mathbf{M}_l = \eta_h \mathbf{I} + \mathbf{\Phi}_l(\mathbf{X})^T \mathbf{A}_l \mathbf{\Phi}_l(\mathbf{X})$, where $\mathbf{I}$ is the identity matrix of dimensionality $d_l \times d_l$, $\mathbf{A}_l$ is a $n \times n$ PSD matrix, $\mathbf{\Phi}_l(\mathbf{X})$ is the matrix of learning instances in the $\mathcal{H}_l$ space (with instances in rows), and $\eta_h$ is a constant that depends on the ML method $h$. Since in the vast majority of the existing ML methods [19, 8, 18, 23, 5, 14, 22] the value of constant $\eta_h$ is zero, in this paper we only consider the optimal form of $\mathbf{M}_l$ with $\eta_h = 0$. Under the optimal parametrization of $\mathbf{M}_l = \mathbf{\Phi}_l(\mathbf{X})^T \mathbf{A}_l \mathbf{\Phi}_l(\mathbf{X})$ the squared Mahalanobis distance becomes:

$$d_{\mathbf{M}_l}^2(\mathbf{\Phi}_l(\mathbf{x}_i), \mathbf{\Phi}_l(\mathbf{x}_j)) \quad = \quad (\mathbf{K}_l^i - \mathbf{K}_l^j)^T \mathbf{A}_l(\mathbf{K}_l^i - \mathbf{K}_l^j) = d_{\mathbf{A}_l}^2(\mathbf{\Phi}_l(\mathbf{x}_i), \mathbf{\Phi}_l(\mathbf{x}_j)) \tag{2}$$

where $\mathbf{K}_l^i$ is the $i$-th column of kernel matrix $\mathbf{K}_l$, the $(i, j)$ element of which is $K_{l_{ij}} = k_l(\mathbf{x}_i, \mathbf{x}_j)$. As a result, (1) can be rewritten as:

$$\min_{\mathbf{A}_l} F_h(\mathbf{\Phi}_l(\mathbf{X})^T \mathbf{A}_l \mathbf{\Phi}_l(\mathbf{X})) \quad s.t. \quad \mathcal{C}_h(d_{\mathbf{A}_l}^2(\mathbf{\Phi}_l(\mathbf{x}_i), \mathbf{\Phi}_l(\mathbf{x}_j))), \ \mathbf{A}_l \succeq 0 \tag{3}$$

In MKL we are given a set of kernel functions $\mathbf{Z} = \{k_l(\mathbf{x}_i, \mathbf{x}_j) \mid l = 1 \ldots m\}$ and the goal is to learn an appropriate kernel function $k_{\boldsymbol{\mu}}(\mathbf{x}_i, \mathbf{x}_j)$ parametrized by $\boldsymbol{\mu}$ under a cost function $Q$. The cost function $Q$ is determined by the cost function of the learning method that is coupled with multiple kernel learning, e.g. it can be the SVM cost function if one is using an SVM as the learning approach. As in [10, 17] we parametrize $k_{\boldsymbol{\mu}}(\mathbf{x}_i, \mathbf{x}_j)$ by a linear combination of the form:

$$k_{\boldsymbol{\mu}}(\mathbf{x}_i, \mathbf{x}_j) = \sum_{i=l}^{m} \mu_l k_l(\mathbf{x}_i, \mathbf{x}_j), \ \mu_l \geq 0, \ \sum_{l}^{m} \mu_l = 1 \tag{4}$$

We denote the feature space that is induced by the $k_{\boldsymbol{\mu}}$ kernel by $\mathcal{H}_{\boldsymbol{\mu}}$, feature space which is given by the mapping $\mathbf{x} \rightarrow \boldsymbol{\Phi}_{\boldsymbol{\mu}}(\mathbf{x}) = (\sqrt{\mu_1} \boldsymbol{\Phi}_1(\mathbf{x})^T, \ldots, \sqrt{\mu_m} \boldsymbol{\Phi}_m(\mathbf{x})^T)^T \in \mathcal{H}_{\boldsymbol{\mu}}$. We denote the dimensionality of $\mathcal{H}_{\boldsymbol{\mu}}$ by $d$; it can be infinite. Finally, we denote by $\mathcal{H}$ the feature space that we get by the unweighted concatenation of the $m$ feature spaces, i.e. $\forall \mu_i, \ \mu_i = 1$, whose representation is given by $\mathbf{x} \rightarrow \boldsymbol{\Phi}(\mathbf{x}) = (\boldsymbol{\Phi}_1(\mathbf{x})^T, \ldots, \boldsymbol{\Phi}_m(\mathbf{x})^T)^T$.

## 3   Metric Learning with Multiple Kernel Learning

The goal is to learn a metric matrix $\mathbf{M}$ in the feature space $\mathcal{H}_{\boldsymbol{\mu}}$ induced by the mapping $\boldsymbol{\Phi}_{\boldsymbol{\mu}}$ as well as the kernel weight $\boldsymbol{\mu}$; we denote this metric by $d_{\mathbf{M}, \boldsymbol{\mu}}^2$. Based on the optimal form of the Mahalanobis metric $\mathbf{M}$ for metric learning method learning with a single kernel function [9], we have the following lemma:

**Lemma 1.** *Assume that for a metric learning method $h$ the optimal parameterization of its Mahalanobis metric $\mathbf{M}^*$ is $\boldsymbol{\Phi}_l(\mathbf{X})^T \mathbf{A}^* \boldsymbol{\Phi}_l(\mathbf{X})$, for some $\mathbf{A}^*$, when learning with a single kernel function $k_l(\boldsymbol{x}, \boldsymbol{x}')$. Then, for $h$ with multiple kernel learning the optimal parametrization of its Mahalanobis metric $\mathbf{M}^{**}$ is given by $\boldsymbol{\Phi}_{\boldsymbol{\mu}}(\mathbf{X})^T \mathbf{A}^{**} \boldsymbol{\Phi}_{\boldsymbol{\mu}}(\mathbf{X})$, for some $\mathbf{A}^{**}$.*

The proof of the above Lemma is similar to the proof of Theorem 1 in [9] (it is not presented here due to the lack of space). Following Lemma 1, we have:

$$
\begin{aligned}
d_{\mathbf{M}, \boldsymbol{\mu}}^2(\boldsymbol{\Phi}_{\boldsymbol{\mu}}(\mathbf{x}_i), \boldsymbol{\Phi}_{\boldsymbol{\mu}}(\mathbf{x}_j)) = & \quad (\boldsymbol{\Phi}_{\boldsymbol{\mu}}(\mathbf{x}_i) - \boldsymbol{\Phi}_{\boldsymbol{\mu}}(\mathbf{x}_j))^T \boldsymbol{\Phi}_{\boldsymbol{\mu}}(\mathbf{X})^T \mathbf{A} \boldsymbol{\Phi}_{\boldsymbol{\mu}}(\mathbf{X})(\boldsymbol{\Phi}_{\boldsymbol{\mu}}(\mathbf{x}_i) - \boldsymbol{\Phi}_{\boldsymbol{\mu}}(\mathbf{x}_j)) \quad (5) \\
= & \quad \sum_l \mu_l(\mathbf{K}_l^i - \mathbf{K}_l^j)^T \mathbf{A} \sum_l \mu_l(\mathbf{K}_l^i - \mathbf{K}_l^j) = d_{\mathbf{A}, \boldsymbol{\mu}}^2(\boldsymbol{\Phi}_{\boldsymbol{\mu}}(\mathbf{x}_i), \boldsymbol{\Phi}_{\boldsymbol{\mu}}(\mathbf{x}_j))
\end{aligned}
$$

Based on (5) and the constraints from (4), the ML optimization problem with MKL can be presented as:

$$\min_{\mathbf{A}, \boldsymbol{\mu}} F_h(\boldsymbol{\Phi}_{\boldsymbol{\mu}}(\mathbf{X})^T \mathbf{A} \boldsymbol{\Phi}_{\boldsymbol{\mu}}(\mathbf{X})) \quad s.t. \quad \mathcal{C}_h(d_{\mathbf{A}, \boldsymbol{\mu}}^2(\boldsymbol{\Phi}_{\boldsymbol{\mu}}(\mathbf{x}_i), \boldsymbol{\Phi}_{\boldsymbol{\mu}}(\mathbf{x}_j))), \ \mathbf{A} \succeq 0, \ \mu_l \geq 0, \ \sum_l^m \mu_l = 1 \tag{6}$$

We denote the resulting optimization problem and the learning method by $ML_h\text{-}MKL_{\boldsymbol{\mu}}$; clearly this is not fully specified until we choose a specific ML method $h$.

Let $\mathbf{B} = \begin{bmatrix} (\mathbf{K}_1^i - \mathbf{K}_1^j)^T \\ \ldots \\ (\mathbf{K}_m^i - \mathbf{K}_m^j)^T \end{bmatrix}$. We note that $d_{\mathbf{A}, \boldsymbol{\mu}}^2(\boldsymbol{\Phi}_{\boldsymbol{\mu}}(\mathbf{x}_i), \boldsymbol{\Phi}_{\boldsymbol{\mu}}(\mathbf{x}_j))$ from (5) can also be written as:

$$d_{\mathbf{A}, \boldsymbol{\mu}}^2(\boldsymbol{\Phi}_{\boldsymbol{\mu}}(\mathbf{x}_i), \boldsymbol{\Phi}_{\boldsymbol{\mu}}(\mathbf{x}_j)) = \boldsymbol{\mu}^T \mathbf{B} \mathbf{A} \mathbf{B}^T \boldsymbol{\mu} = tr(\mathbf{P} \mathbf{B} \mathbf{A} \mathbf{B}^T) = d_{\mathbf{A}, \mathbf{P}}^2(\boldsymbol{\Phi}_{\mathbf{P}}(\mathbf{x}_i), \boldsymbol{\Phi}_{\mathbf{P}}(\mathbf{x}_j)) \tag{7}$$

where $\mathbf{P} = \boldsymbol{\mu} \boldsymbol{\mu}^T$ and $tr(\cdot)$ is the trace of a matrix. We use $\boldsymbol{\Phi}_{\mathbf{P}}(\mathbf{X})$ to emphasize the explicit the dependence of $\boldsymbol{\Phi}_{\boldsymbol{\mu}}(\mathbf{X})$ to $\mathbf{P} = \boldsymbol{\mu} \boldsymbol{\mu}^T$. As a result, instead of optimizing over $\boldsymbol{\mu}$ we can also use the parametrization over $\mathbf{P}$; the new optimization problem can now be written as:

$$\min_{\mathbf{A}, \mathbf{P}} \quad F_h(\boldsymbol{\Phi}_{\mathbf{P}}(\mathbf{X})^T \mathbf{A} \boldsymbol{\Phi}_{\mathbf{P}}(\mathbf{X})) \tag{8}$$

$$s.t. \quad \mathcal{C}_h(d_{\mathbf{A}, \mathbf{P}}^2(\boldsymbol{\Phi}_{\mathbf{P}}(\mathbf{x}_i), \boldsymbol{\Phi}_{\mathbf{P}}(\mathbf{x}_j))), \ \mathbf{A} \succeq 0, \ \sum_{ij} P_{ij} = 1, \ P_{ij} \geq 0, \ Rank(\mathbf{P}) = 1, \ \mathbf{P} = \mathbf{P}^T$$

where the constraints $\sum_{ij} P_{ij} = 1$, $P_{ij} \geq 0$, $Rank(\mathbf{P}) = 1$, and $\mathbf{P} = \mathbf{P}^T$ are added so that $\mathbf{P} = \boldsymbol{\mu} \boldsymbol{\mu}^T$. We call the optimization problem and learning method (8) as $ML_h\text{-}MKL_{\mathbf{P}}$; as before in order to fully instantiate it we need to choose a specific metric learning method $h$.

Now, we derive an alternative parametrization of (5). We need two additional matrices: $\mathbf{C}_{\mu_i \mu_j} = \mu_i \mu_j \mathbf{I}$, where the dimensionality of $\mathbf{I}$ is $n \times n$, and $\mathbf{\Phi}^{'}(\mathbf{X})$ which is an $mn \times d$ dimensional matrix:

$$\mathbf{\Phi}^{'}(\mathbf{X}) = \left[ \begin{array}{ccc} \mathbf{\Phi}_1(\mathbf{X}) & \ldots & \mathbf{0} \\ \ldots & \ldots & \ldots \\ \mathbf{0} & \ldots & \mathbf{\Phi}_m(\mathbf{X}) \end{array} \right]$$

We have:

$$d^2_{\mathbf{A},\boldsymbol{\mu}}(\mathbf{\Phi}_{\boldsymbol{\mu}}(\mathbf{x}_i), \mathbf{\Phi}_{\boldsymbol{\mu}}(\mathbf{x}_j)) = (\mathbf{\Phi}(\mathbf{x}_i) - \mathbf{\Phi}(\mathbf{x}_j))^T \mathbf{M}^{'}(\mathbf{\Phi}(\mathbf{x}_i) - \mathbf{\Phi}(\mathbf{x}_j)) \tag{9}$$

where:

$$\mathbf{M}^{'} = \mathbf{\Phi}^{'}(\mathbf{X})^T \mathbf{A}^{'} \mathbf{\Phi}^{'}(\mathbf{X}) \tag{10}$$

and $\mathbf{A}^{'}$ is a $mn \times mn$ matrix:

$$\mathbf{A}^{'} = \left[ \begin{array}{ccc} \mathbf{C}_{\mu_1 \mu_1} \mathbf{A} & \ldots & \mathbf{C}_{\mu_1 \mu_m} \mathbf{A} \\ \ldots & \ldots & \ldots \\ \mathbf{C}_{\mu_m \mu_1} \mathbf{A} & \ldots & \mathbf{C}_{\mu_m \mu_m} \mathbf{A} \end{array} \right]. \tag{11}$$

From (9) we see that the Mahalanobis metric, parametrized by the $\mathbf{M}$ or $\mathbf{A}$ matrix, in the feature space $\mathcal{H}_{\boldsymbol{\mu}}$ induced by the kernel $k_{\boldsymbol{\mu}}$, is equivalent to the Mahalanobis metric in the feature space $\mathcal{H}$ which is parametrized by $\mathbf{M}^{'}$ or $\mathbf{A}^{'}$. As we can see from (11), $ML_h\text{-}MKL_{\boldsymbol{\mu}}$ and $ML_h\text{-}MKL_{\mathbf{P}}$ learn a *regularized* matrix $\mathbf{A}^{'}$ (i.e. matrix with internal structure) that corresponds to a parametrization of the Mahalanobis metric $\mathbf{M}^{'}$ in the feature space $\mathcal{H}$.

### 3.1 Non-Regularized Metric Learning with Multiple Kernel Learning

We present here a more general formulation of the optimization problem (6) in which we lift the regularization of matrix $\mathbf{A}^{'}$ from (11), and learn instead a full PSD matrix $\mathbf{A}^{''}$:

$$\mathbf{A}^{''} = \left[ \begin{array}{ccc} \mathbf{A}_{11} & \ldots & \mathbf{A}_{1m} \\ \ldots & \ldots & \ldots \\ \mathbf{A}_{1m} & \ldots & \mathbf{A}_{mm} \end{array} \right] \tag{12}$$

where $\mathbf{A}_{kl}$ is an $n \times n$ matrix. The respective Mahalanobis matrix, which we denote by $\mathbf{M}^{''}$, still have the same parametrization form as in (10), i.e. $\mathbf{M}^{''} = \mathbf{\Phi}^{'}(\mathbf{X})^T \mathbf{A}^{''} \mathbf{\Phi}^{'}(\mathbf{X})$. As a result, by using $\mathbf{A}^{''}$ instead of $\mathbf{A}^{'}$ the squared Mahalanobis distance can be written now as:

$$\begin{aligned} d^2_{\mathbf{A}^{''}}(\mathbf{\Phi}(\mathbf{x}_i), \mathbf{\Phi}(\mathbf{x}_j)) &= (\mathbf{\Phi}(\mathbf{x}_i) - \mathbf{\Phi}(\mathbf{x}_j))^T \mathbf{M}^{''}(\mathbf{\Phi}(\mathbf{x}_i) - \mathbf{\Phi}(\mathbf{x}_j)) \\ &= [(\mathbf{K}_1^i - \mathbf{K}_1^j)^T, \ldots, (\mathbf{K}_m^i - \mathbf{K}_m^j)^T] \mathbf{A}^{''} [(\mathbf{K}_1^i - \mathbf{K}_1^j)^T, \ldots, (\mathbf{K}_m^i - \mathbf{K}_m^j)^T]^T \\ &= [\mathbf{\Phi}_{\mathbf{Z}}(\mathbf{x}_i) - \mathbf{\Phi}_{\mathbf{Z}}(\mathbf{x}_j)]^T \mathbf{A}^{''} (\mathbf{\Phi}_{\mathbf{Z}}(\mathbf{x}_i) - \mathbf{\Phi}_{\mathbf{Z}}(\mathbf{x}_j)) \end{aligned} \tag{13}$$

where $\mathbf{\Phi}_{\mathbf{Z}}(\mathbf{x}_i) = ((\mathbf{K}_1^i)^T, \ldots, (\mathbf{K}_m^i)^T)^T \in \mathcal{H}_{\mathbf{Z}}$. What we see here is that under the $\mathbf{M}^{''}$ parametrization computing the Mahalanobis metric in the $\mathcal{H}$ is equivalent to computing the Mahalanobis metric in the $\mathcal{H}_{\mathbf{Z}}$ space. Under the parametrization of the Mahalanobis distance given by (13), the optimization problem of metric learning with multiple kernel learning is the following:

$$\min_{\mathbf{A}^{''}} F_h(\mathbf{\Phi}^{'}(\mathbf{X})^T \mathbf{A}^{''} \mathbf{\Phi}^{'}(\mathbf{X})) \quad s.t. \quad \mathcal{C}_h(d^2_{\mathbf{A}^{''}}(\mathbf{\Phi}(\mathbf{x}_i), \mathbf{\Phi}(\mathbf{x}_j))), \ \mathbf{A}^{''} \succeq 0 \tag{14}$$

We call this optimization problem *NR-ML_h-MKL*. We should note that this formulation has scaling problems since it has $O(m^2 n^2)$ parameters that need to be estimated, and it clearly requires a very efficient ML method $h$ in order to be practical.

## 4 Optimization

### 4.1 Analysis

The *NR-ML_h-MKL* optimization problem obviously has the same convexity properties as the metric learning algorithm $h$ that will be used, since the parametrization $\mathbf{M}^{''} = \mathbf{\Phi}^{'}(\mathbf{X})^T \mathbf{A}^{''} \mathbf{\Phi}^{'}(\mathbf{X})$ used in *NR-ML_h-MKL* is linear with $\mathbf{A}^{''}$, and the composition of a function with an affine mapping preserves

the convexity property of the original function [1]. This is also valid for the subproblems of learning matrix $\mathbf{A}$ in $ML_h$-$MKL_{\boldsymbol{\mu}}$ and $ML_h$-$MKL_{\mathbf{P}}$ given the weight vector $\boldsymbol{\mu}$.

Given the PSD matrix $\mathbf{A}$, we have the following two lemmas for optimization problems $ML_h$-$MKL_{\{\boldsymbol{\mu}|\mathbf{P}\}}$:

**Lemma 2.** *Given the PSD matrix $\mathbf{A}$ the* $\mathrm{ML}_h$-$\mathrm{MKL}_{\boldsymbol{\mu}}$ *optimization problem is convex with $\boldsymbol{\mu}$ if metric learning algorithm $h$ is convex with $\boldsymbol{\mu}$.*

*Proof.* The last two constraints on $\boldsymbol{\mu}$ of the optimization problem from (6) are linear, thus this problem is convex if metric learning algorithm $h$ is convex with $\boldsymbol{\mu}$. $\qquad\square$

Since $d^2_{\mathbf{A},\boldsymbol{\mu}}(\boldsymbol{\Phi}_{\boldsymbol{\mu}}(\mathbf{x}_i), \boldsymbol{\Phi}_{\boldsymbol{\mu}}(\mathbf{x}_j))$ is convex quadratic of $\boldsymbol{\mu}$, which can be easily proved based on the PSD property of matrix $\mathbf{BAB^T}$ in (7), many of the well known metric learning algorithms, such as Pairwise SVM [21], POLA [19] and Xing's method [23] satisfy the conditions in Lemma 2.

The $ML_h$-$MKL_{\mathbf{P}}$ optimization problem (8) is not convex given a PSD matrix $\mathbf{A}$ because the rank constraint is not convex. However, when the number of kernels $m$ is small, e.g. a few tens of kernels, there is an equivalent convex formulation.

**Lemma 3.** *Given the PSD matrix $\mathbf{A}$, the* $\mathrm{ML}_h$-$\mathrm{MKL}_{\mathbf{P}}$ *optimization problem (8) can be formulated as an equivalent convex problem with respect to $\mathbf{P}$ if the ML algorithm $h$ is linear with $\mathbf{P}$ and the number of kernel $m$ is small.*

*Proof.* Given the PSD matrix $\mathbf{A}$, if $h$ is linear with $\mathbf{P}$, we can formulate the rank constraint problem with the help of the two following convex problems [2]:

$$\min_{\mathbf{P}} \quad F_h(\boldsymbol{\Phi}_{\mathbf{P}}(\mathbf{X})^T \mathbf{A} \boldsymbol{\Phi}_{\mathbf{P}}(\mathbf{X})) + w \cdot tr(\mathbf{P}^T \mathbf{W}) \tag{15}$$

$$s.t. \quad \mathcal{C}_h(d^2_{\mathbf{A},\mathbf{P}}(\boldsymbol{\Phi}_{\mathbf{P}}(\mathbf{x}_i),\ \boldsymbol{\Phi}_{\mathbf{P}}(\mathbf{x}_j))),\ \mathbf{A} \succeq 0,\ \mathbf{P} \succeq 0,\ \sum_{ij} P_{ij} = 1,\ P_{ij} \geq 0,\ \mathbf{P} = \mathbf{P}^T$$

where $w$ is a positive scalar just enough to make $tr(\mathbf{P}^T \mathbf{W})$ vanish, i.e. global convergence defined in (17), and the direction matrix $\mathbf{W}$ is an optimal solution of the following problem:

$$\min_{\mathbf{W}} tr(\mathbf{P}^{*T} \mathbf{W}) \quad s.t. \quad \mathbf{0} \preceq \mathbf{W} \preceq \mathbf{I},\ tr(\mathbf{W}) = m - 1 \tag{16}$$

where $\mathbf{P}^*$ is an optimal solution of (15) given $\mathbf{A}$ and $\mathbf{W}$, and $m$ is the number of kernels. The problem (16) has a closed form solution $\mathbf{W} = \mathbf{UU}^T$, where $\mathbf{U} \in \mathbb{R}^{m \times m-1}$ is the eigenvector matrix of $\mathbf{P}^*$ whose columns are the eigenvectors which correspond to the $m - 1$ smallest eigenvalues of $\mathbf{P}^*$. The two convex problems are iteratively solved until global convergence, defined as:

$$\sum_{i=2}^{m} \lambda(\mathbf{P}^*)_i = tr(\mathbf{P}^{*T} \mathbf{W}^*) = \lambda(\mathbf{P}^*)^T \lambda(\mathbf{W}^*) \equiv 0 \tag{17}$$

where $\lambda(\mathbf{P}^*)_i$ is the $i$-th largest eigenvalue of $\mathbf{P}^*$. This formulation is not a projection method. At global convergence the convex problem (15) is not a relaxation of the original problem, instead it is an equivalent convex problem [2].

We will now prove the convergence of problem (15). Suppose the objective value of (15) is $f_i$ at iteration $i$. Since both (15) and (16) minimize the objective value of (15), we have $f_j < f_i$ for any iteration $j > i$. Beacuse the infimum $f^*$ of the objective value of (15) corresponds to the optimal objective value of (15) when the second term is removed. Thus the nonincreasing sequence of objective values is bounded below and as a result converges because any bounded monotonic sequence in $\mathbb{R}$ is convergent. Thus the local convergence of (15) is now established.

Only the local convergence can be established for problem (15) because the objective $tr(\mathbf{P}^T \mathbf{W})$ is generally multimodal [2]. However, as indicated in section 7.2 [2], when the size of $m$ is small, the global optimal of problem (15) can be often achieved. This can be simply verified by comparing the difference between the infimum $f^*$ and the optimal objective value $f$ of problem (15). $\qquad\square$

For a number of known metric learning algorithms, such as LMNN [22], POLA [19], MLSVM [14] and Xing's method [23] linearity with respect to $\mathbf{P}$ holds given $\mathbf{A} \succeq 0$.

**Algorithm 1** $ML_h$-$MKL_{\boldsymbol{\mu}}$, $ML_h$-$MKL_{\mathbf{P}}$

---

**Input:** $\mathbf{X}$, $\mathbf{Y}$, $\mathbf{A}^0$, $\boldsymbol{\mu}^0$, and matrices $\mathbf{K}_1, \ldots, \mathbf{K}_m$
**Output:** $\mathbf{A}$ and $\boldsymbol{\mu}$
**repeat**
  $\boldsymbol{\mu}^{(i)}$=WeightLearning($\mathbf{A}^{(i-1)}$)
  $\mathbf{K}_{\boldsymbol{\mu}^{(i)}} = \sum_k \mu_k^i \mathbf{K}_k$
  $\mathbf{A}^{(i)}$=MetricLearning$_h$($\mathbf{A}^{(i-1)}$,$\mathbf{X}$,$\mathbf{K}_{\boldsymbol{\mu}^{(i)}}$)
  $i := i + 1$
**until** convergence

---

## 4.2 Optimization Algorithms

The *NR-$ML_h$-MKL* optimization problem can be directly solved by any metric learning algorithm $h$ on the space $\mathcal{H}_{\mathbf{Z}}$ when the optimization problem of the latter only involves the squared pairwise Mahalanobis distance, e.g. LMNN [22] and MCML [5]. When the metric learning algorithm $h$ has regularization term on $\mathbf{M}$, e.g. trace norm [8] and Frobenius norm [14, 19], most often the *NR-$ML_h$-MKL* optimization problem can be solved by a slightly modification of original algorithm.

We now describe how we can solve the optimization problems of $ML_h$-$MKL_{\boldsymbol{\mu}}$ and $ML_h$-$MKL_{\mathbf{P}}$. Based on Lemmas 2 and 3 we propose for both methods a two-step iterative algorithm, Algorithm 1, at the first step of which we learn the kernel weighting and at the second the metric under the kernel weighting learned in the first step. At the first step of the $i$-th iteration we learn the $\boldsymbol{\mu}^{(i)}$ kernel weight vector under fixed PSD matrices $\mathbf{A}^{(i-1)}$, learned at the preceding iteration $(i-1)$. For $ML_h$-$MKL_{\boldsymbol{\mu}}$ we solve the weight learning problem using Lemma 2 and for $ML_h$-$MKL_{\mathbf{P}}$ using Lemma 3. At the second step we apply the metric learning algorithm $h$ and we learn the PSD matrices $\mathbf{A}^{(i)}$ with the $\mathbf{K}_{\boldsymbol{\mu}^{(i)}} = \sum_l \mu_l^{(i)} \mathbf{K}_i$ kernel matrix using as the initial metric matrices the $\mathbf{A}^{(i-1)}$. We should make clear that the optimization problem we are solving is only individually convex with respect to $\boldsymbol{\mu}$ given the PSD matrix $\mathbf{A}$ and vice-versa. As a result, the convergence of the two-step algorithm (possible to a local optima) is guaranteed [6] and checked by the variation of $\boldsymbol{\mu}$ and the objective value of metric learning method $h$. In our experiments (Section 6) we observed that it most often converges in less than ten iterations.

## 5 *LMNN*-Based Instantiation

We have presented two basic approaches to metric learning with multiple kernel learning: $ML_h$-$MKL_{\boldsymbol{\mu}}$ ($ML_h$-$MKL_{\mathbf{P}}$) and *NR-$ML_h$-MKL*. In order for the approaches to be fully instantiated we have to specify the ML algorithm $h$. In this paper we focus on the LMNN state-of-the-art method [22].

Due to the relative comparison constraint, LMNN does not satisfy the condition of Lemma 2. However, as we already mentioned LMNN satisfies the condition of Lemma 3 so we get the $ML_h$-$MKL_{\mathbf{P}}$ variant of the optimization problem for LMNN which we denote by *LMNN-$MKL_{\mathbf{P}}$*. The resulting optimization problem is:

$$\min_{\mathbf{A},\mathbf{P},\xi} \quad \sum_{ij} \mathbf{S}_{ij}\{(1-\gamma)d_{\mathbf{A},\mathbf{P}}^2(\boldsymbol{\Phi}_{\mathbf{P}}(\mathbf{x}_i), \boldsymbol{\Phi}_{\mathbf{P}}(\mathbf{x}_j)) + \gamma\sum_k(1-\mathbf{Y}_{ik})\xi_{ijk}\} \tag{18}$$

$$s.t. \quad d_{\mathbf{A},\mathbf{P}}^2(\boldsymbol{\Phi}_{\mathbf{P}}(\mathbf{x}_i), \boldsymbol{\Phi}_{\mathbf{P}}(\mathbf{x}_k)) - d_{\mathbf{A},\mathbf{P}}^2(\boldsymbol{\Phi}_{\mathbf{P}}(\mathbf{x}_i), \boldsymbol{\Phi}_{\mathbf{P}}(\mathbf{x}_j)) \geq 1 - \xi_{ijk}, \; \xi_{ijk} > 0, \; \mathbf{A} \succeq 0$$

$$\sum_{kl} P_{kl} = 1, \; P_{kl} \geq 0, \; Rank(\mathbf{P}) = 1, \; \mathbf{P} = \mathbf{P}^T$$

where the matrix $\mathbf{Y}$, $\mathbf{Y}_{ij} \in \{0, 1\}$, indicates if the class labels $y_i$ and $y_j$ are the same ($\mathbf{Y}_{ij} = 1$) or different ($\mathbf{Y}_{ij} = 0$). The matrix $\mathbf{S}$ is a binary matrix whose $S_{ij}$ entry is non-zero if instance $\mathbf{x}_j$ is one of the $k$ same class nearest neigbors of instance $\mathbf{x}_i$. The objective is to minimize the sum of the distances of all instances to their $k$ same class nearest neighbors while allowing for some errors, trade of which is controlled by the $\gamma$ parameter. As the objective function of LMNN only involves the squared pairwise Mahalanobis distances, the instantiation of *NR-$ML_h$-MKL* is straightforward and it consists simply of the application of LMNN on the space $\mathcal{H}_{\mathbf{Z}}$ in order to learn the metric. We denote this instantiation by *NR-LMNN-MKL*.

Table 1: Accuracy results. The superscripts $^{+-=}$ next to the accuracies of *NR-LMNN-MKL* and *LMNN-MKL*$_\mathbf{P}$ indicate the result of the McNemar's statistical test of their comparison to the accuracies of *LMNN*$_\mathcal{H}$ and *LMNN-MKL*$_{CV}$ and denote respectively a significant win, loss or no difference. The number in the parenthesis indicates the score of the respective algorithm for the given dataset based on the pairwise comparisons of the McNemar's statistical test.

| Datasets | NR-LMNN-MKL | LMNN-MKL$_\mathbf{P}$ | LMNN$_\mathcal{H}$ | LMNN-MKL$_{CV}$ | 1-NN |
|---|---|---|---|---|---|
| Sonar | $88.46^{+=}$(3.0) | $85.58^{==}$(2.0) | 82.21(1.0) | 88.46(3.0) | 82.21(1.0) |
| Wine | $98.88^{==}$(2.0) | $98.88^{==}$(2.0) | 98.31(2.0) | 96.07(2.0) | 97.19(2.0) |
| Iris | $93.33^{==}$(2.0) | $95.33^{==}$(2.0) | 94.67(2.0) | 94.00(2.0) | 95.33(2.0) |
| Ionosphere | $93.73^{==}$(2.5) | $94.87^{=+}$(3.0) | 92.59(2.5) | 90.88(2.0) | 86.89(0.0) |
| Wdbc | $94.90^{-=}$(1.0) | $97.36^{=+}$(3.5) | 97.36(3.0) | 95.96(1.5) | 95.43(1.0) |
| CentralNervous | $55.00^{==}$(2.0) | $63.33^{==}$(2.0) | 65.00(2.0) | 65.00(2.0) | 58.33(2.0) |
| Colon | $80.65^{==}$(2.0) | $85.48^{+=}$(2.5) | 66.13(1.5) | 79.03(2.0) | 74.19(2.0) |
| Leukemia | $95.83^{+=}$(2.5) | $94.44^{+=}$(2.5) | 70.83(0.0) | 95.83(2.5) | 88.89(2.5) |
| MaleFemale | $86.57^{==}$(2.5) | $88.81^{+=}$(3.0) | 80.60(1.5) | 89.55(2.5) | 58.96(0.0) |
| Ovarian | $95.26^{+=}$(3.0) | $94.47^{+=}$(3.0) | 90.51(0.5) | 94.47(3.0) | 87.35(0.5) |
| Prostate | $79.50^{==}$(2.0) | $80.43^{==}$(2.5) | 79.19(2.0) | 78.88(2.0) | 76.71(1.5) |
| Stroke | $69.71^{==}$(2.0) | $72.12^{==}$(2.0) | 71.15(2.0) | 70.19(2.0) | 65.38(2.0) |
| Total Score | 26.5 | 30.0 | 20.0 | 27.0 | 16.5 |

## 6 Experiments

In this section we perform a number of experiments on real world datasets in order to compare the two of the LMNN-based instantiations of our framework, i.e. *LMNN-MKL*$_\mathbf{P}$ and *NR-LMNN-MKL*. We compare these methods against two baselines: *LMNN-MKL*$_{CV}$ in which a kernel is selected from a set of kernels using 2-fold inner cross-validation (CV), and LMNN with the unweighted sum of kernels, which induces the $\mathcal{H}$ feature space, denoted by *LMNN*$_\mathcal{H}$. Additionally, we report performance of 1-Nearest-Neighbor, denoted as 1-NN, with no metric learning. The PSD matrix $\mathbf{A}$ and weight vector $\boldsymbol{\mu}$ in *LMNN-MKL*$_\mathbf{P}$ were respectively initialized by $\mathbf{I}$ and equal weighting (1 divided by the number of kernels). The parameter $w$ in the weight learning subproblem of *LMNN-MKL*$_\mathbf{P}$ was selected from $\{10^i \mid i = 0, 1, \ldots, 8\}$ and was the smallest value enough to achieve global convergence. Its direction matrix $\mathbf{W}$ was initialized by $\mathbf{0}$. The number of $k$ same class nearest neighbors required by LMNN was set to 5 and its $\gamma$ parameter to 0.5. After learning the metric and the multiple kernel combination we used 1-NN for classification.

### 6.1 Benchmark Datasets

We first experimented with 12 different datasets: five from the UCI machine learning repository, i.e. Sonar, Ionosphere, Wine, Iris, and Wdbc; three microarray datasets, i.e. CentralNervous, Colon, and Leukemia; and four proteomics datasets, i.e. MaleFemale, Stroke, Prostate and Ovarian. The attributes of all the datasets are standardized in the preprocessing step. The $\mathbf{Z}$ set of kernels that we use consists of the following 20 kernels: 10 polynomial with degree from one to ten, ten Gaussians with bandwidth $\sigma \in \{0.5, 1, 2, 5, 7, 10, 12, 15, 17, 20\}$ (the same set of kernels was used in [4]). Each basic kernel $\mathbf{K}_k$ was normalized by the average of its $diag(\mathbf{K}_k)$. *LMNN-MKL*$_\mathbf{P}$, *LMNN*$_\mathcal{H}$ and *LMNN-MKL*$_{CV}$ were tested using the complete $\mathbf{Z}$ set. For *NR-LMNN-MKL* due to its scaling limitations we could only use a small subset of $\mathbf{Z}$ consisting of the linear, the second order polynomial, and the Gaussian kernel with the kernel width of 0.5. We use 10-fold CV to estimate the predictive performance of the different methods. To test the statistical significance of the differences we used McNemar's test and we set the p-value to 0.05. To get a better understanding of the relative performance of the different methods for a given dataset we used a ranking schema in which a method A was assigned one point if its accuracy was significantly better than that of another method B, 0.5 points if the two methods did not have a significantly different performance, and zero points if A was found to be significantly worse than B.

The results are reported in Table 1. First, we observe that by learning the kernel inside *LMNN-MKL*$_\mathbf{P}$ we improve performance over *LMNN*$_\mathcal{H}$ that uses the unweighted kernel combination. More precisely, *LMNN-MKL*$_\mathbf{P}$ is significantly better than *LMNN*$_\mathcal{H}$ in four out of the thirteen datasets. If we now compare *LMNN-MKL*$_\mathbf{P}$ with *LMNN-MKL*$_{CV}$, the other baseline method where we select the best kernel with CV, we can see that *LMNN-MKL*$_\mathbf{P}$ also performs better being statistically significant

Table 2: Accuracy results on the multiple source datasets.

| Datasets | $LMNN\text{-}MKL_{\mathbf{P}}$ | $LMNN_{\mathcal{H}}$ | $LMNN\text{-}MKL_{CV}$ | 1-NN |
|---|---|---|---|---|
| Multiple Feature | $98.79^{++}(3.0)$ | 98.44(1.5) | 98.44(1.5) | 97.86(0.0) |
| Oxford Flowers | $86.01^{++}(3.0)$ | 85.74(2.0) | 65.46(0.0) | 67.38(1.0) |

better in two dataset. If we now examine *NR-LMNN-MKL* and *LMNN*$_{\mathcal{H}}$ we see that the former method, even though learning with only three kernels, is significantly better in two datasets, while it is significantly worse in one dataset. Comparing *NR-LMNN-MKL* and *LMNN-MKL*$_{CV}$ we observe that the two methods achieve comparable predictive performances. We should stress here that *NR-LMNN-MKL* has a disadvantage since it only uses three kernels, as opposed to other methods that use 20 kernels; the scalability of *NR-LMNN-MKL* is left as a future work. In terms of the total score that the different methods obtain the best one is *LMNN-MKL*$_{\mathbf{P}}$ followed by *LMNN-MKL*$_{CV}$ and *NR-LMNN-MKL*.

## 6.2 Multiple Source Datasets

To evaluate the proposed method on problems with multiple sources of information we also perform experiments on the Multiple Features and the Oxford flowers datasets [16]. Multiple Features from UCI has six different feature representations for 2,000 handwritten digits (0-9); each class has 200 instances. In the preprocessing step all the features are standardized in all the data sources. Oxford flowers dataset has 17 category flower images; each class has 80 instances. In the experiment seven distance matrices from the website[2] are used; these matrices are precomputed respectively from seven features, the details of which are described in [16, 15]. For both datasets Gaussian kernels are constructed respectively using the different feature representations of instances with kernel width $\sigma_0$, where $\sigma_0$ is the mean of all pairwise distances. We experiment with 10 random splits where half of the data is used for training and the other half for testing. We do not experiment here with *NR-LMNN-MKL* here due to its scaling limitations.

The accuracy results are reported in Table 2. We can see that by learning a linear combination of different feature representations *LMNN-MKL*$_{\mathbf{P}}$ achieves the best predictive performance on both datasets being significantly better than the two baselines, *LMNN*$_{\mathcal{H}}$ and *LMNN-MKL*$_{CV}$. The bad performance of *LMNN-MKL*$_{CV}$ on the Oxford flowers dataset could be explained by the fact that the different Gaussian kernels are complementary for the given problem, but in *LMNN-MKL*$_{CV}$ only one kernel is selected.

## 7 Conclusions

In this paper we combine two recent developments in the field of machine learning, namely metric learning and multiple kernel learning, and propose a general framework for learning a metric in a feature space induced by a weighted combination of a number of individual kernels. This is in contrast with the existing kernelized metric learning techniques which consider only one kernel function (or possibly an unweighted combination of a number of kernels) and hence are sensitive to the selection of the associated feature space. The proposed framework is general as it can be coupled with many existing metric learning techniques. In this work, to practically demonstrate the effectiveness of the proposed approach, we instantiate it with the well know LMNN metric learning method. The experimental results confirm that the adaptively induced feature space does bring an advantage in the terms of predictive performance with respect to feature spaces induced by an unweighted combination of kernels and the single best kernel selected by internal CV.

**Acknowledgments**

This work was funded by the Swiss NSF (Grant 200021-122283/1). The support of the European Commission through EU projects DebugIT (FP7-217139) and e-LICO (FP7-231519) is also gratefully acknowledged.

## Footnotes

[1]The optimization could also be done with respect to other variables of the cost function and not only $\mathbf{M}_l$. However, to keep the notation uncluttered we parametrize the optimization problem only over $\mathbf{M}_l$.

[2]http://www.robots.ox.ac.uk/∼vgg/data/flowers/index.html

# References

[1] S.P. Boyd and L. Vandenberghe. *Convex optimization*. Cambridge Univ Pr, 2004.

[2] J. Dattorro. *Convex optimization & Euclidean distance geometry*. Meboo Publishing USA, 2005.

[3] J.V. Davis, B. Kulis, P. Jain, S. Sra, and I.S. Dhillon. Information-theoretic metric learning. In *ICML*, 2007.

[4] K. Gai, G. Chen, and C. Zhang. Learning kernels with radiuses of minimum enclosing balls. *NIPS*, 2010.

[5] A. Globerson and S. Roweis. Metric learning by collapsing classes. In *NIPS*, 2006.

[6] L. Grippo and M. Sciandrone. On the convergence of the block nonlinear gauss-seidel method under convex constraints* 1. *Operations Research Letters*, 26(3):127–136, 2000.

[7] M. Guillaumin, J. Verbeek, and C. Schmid. Is that you? Metric learning approaches for face identification. In *ICCV*, pages 498–505, 2009.

[8] K. Huang, Y. Ying, and C. Campbell. Gsml: A unified framework for sparse metric learning. In *Data Mining, 2009. ICDM'09. Ninth IEEE International Conference on*, pages 189–198. IEEE, 2009.

[9] P. Jain, B. Kulis, and I. Dhillon. Inductive regularized learning of kernel functions. *NIPS*, 2010.

[10] G.R.G. Lanckriet, N. Cristianini, P. Bartlett, L. El Ghaoui, and M.I. Jordan. Learning the Kernel Matrix with Semidefinite Programming. *Journal of Machine Learning Research*, 5:27–72, 2004.

[11] B. McFee and G. Lanckriet. Partial order embedding with multiple kernels. In *Proceedings of the 26th Annual International Conference on Machine Learning*, pages 721–728. ACM, 2009.

[12] B. McFee and G. Lanckriet. Metric learning to rank. In *ICML*. ACM New York, NY, USA, 2010.

[13] B. McFee and G. Lanckriet. Learning multi-modal similarity. *The Journal of Machine Learning Research*, 12:491–523, 2011.

[14] N. Nguyen and Y. Guo. Metric Learning: A Support Vector Approach. In *ECML/PKDD*, 2008.

[15] M.E. Nilsback and A. Zisserman. A visual vocabulary for flower classification. In *Computer Vision and Pattern Recognition, 2006 IEEE Computer Society Conference on*, volume 2, pages 1447–1454. Ieee, 2006.

[16] M.E. Nilsback and A. Zisserman. Automated flower classification over a large number of classes. In *Computer Vision, Graphics & Image Processing, 2008. ICVGIP'08. Sixth Indian Conference on*, pages 722–729. IEEE, 2008.

[17] A. Rakotomamonjy, F. Bach, S. Canu, and Y. Grandvalet. SimpleMKL. *Journal of Machine Learning Research*, 9:2491–2521, 2008.

[18] M. Schultz and T. Joachims. Learning a distance metric from relative comparisons. In *NIPS*, 2003.

[19] S. Shalev-Shwartz, Y. Singer, and A.Y. Ng. Online and batch learning of pseudo-metrics. In *Proceedings of the twenty-first international conference on Machine learning*. ACM, 2004.

[20] S. Sonnenburg, G. Ratsch, and C. Schafer. A general and efficient multiple kernel learning algorithm. In *NIPS*, 2006.

[21] J.P. Vert, J. Qiu, and W. Noble. A new pairwise kernel for biological network inference with support vector machines. *BMC bioinformatics*, 8(Suppl 10):S8, 2007.

[22] K.Q. Weinberger and L.K. Saul. Distance metric learning for large margin nearest neighbor classification. *The Journal of Machine Learning Research*, 10:207–244, 2009.

[23] E.P. Xing, A.Y. Ng, M.I. Jordan, and S. Russell. Distance metric learning with application to clustering with side-information. In *NIPS*, 2003.

